# A spike based learning neuron in analog VLSI

**Philipp Häfliger**
Institute of Neuroinformatics
ETHZ/UNIZ
Gloriastrasse 32
CH-8006 Zürich
Switzerland
e-mail: hafliger@neuroinf.ethz.ch
tel: ++41 1 257 26 84

**Misha Mahowald**
Institute of Neuroinformatics
ETHZ/UNIZ
Gloriastrasse 32
CH-8006 Zürich
Switzerland
e-mail: misha@neuroinf.ethz.ch
tel: ++41 1 257 26 84

**Lloyd Watts**
Arithmos, Inc.
2730 San Tomas Expressway, Suite 210
Santa Clara, CA 95051-0952
USA
e-mail: lloyd@arithmos.com
tel: 408 982 4490, x219

## Abstract

Many popular learning rules are formulated in terms of continuous, analog inputs and outputs. Biological systems, however, use action potentials, which are digital-amplitude events that encode analog information in the inter-event interval. Action-potential representations are now being used to advantage in neuromorphic VLSI systems as well. We report on a simple learning rule, based on the Riccati equation described by Kohonen [1], modified for action-potential neuronal outputs. We demonstrate this learning rule in an analog VLSI chip that uses volatile capacitive storage for synaptic weights. We show that our time-dependent learning rule is sufficient to achieve approximate weight normalization and can detect temporal correlations in spike trains.

# 1   INTRODUCTION

It is an ongoing debate how information in the nervous system is encoded and carried between neurons. In many subsystems of the brain it is now believed that it is done by the exact timing of spikes. Furthermore spike signals on VLSI chips allow the use of address-event busses to solve the problem of the large connectivity in neural networks [3, 4]. For these reasons our artificial neuron and others [2] use spike signals to communicate. Additionally the weight updates at the synapses are determined by the relative timing of presynaptic and postsynaptic spikes, a mechanism that has recently been discovered to operate in cortical synapses [5, 7, 6].

Weight normalization is a useful property of learning rules. In order to perform the normalization, some information about the whole weight vector must be available at every synapse. We use the neuron's output spikes (The neuron's output is the product of the weight and the input vector), which retrogradely propagate through the dendrites to the synapses (as has been observed in biological neurons [5]). In our model approximate normalization is an implicit property of the learning rule.

# 2   THE LEARNING RULE

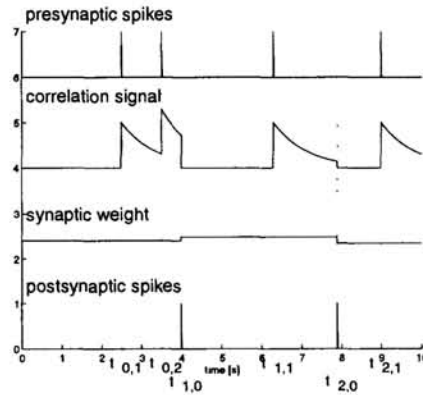

Figure 1: A snapshot of the simulation variables involved at one synapse. With $\tau = 0.83s$

The core of the learning rule is a local 'correlation signal' $c$ at every synapse. It records the 'history' of presynaptic spikes. It is incremented by 1 with every presynaptic spike and decays in time with time constant $\tau$:

$$c(t_{m,0}) = 0$$
$$c(t_{m,n}) = e^{-\frac{t_{m,n}-t_{m,n-1}}{\tau}} c(t_{m,n-1}) + 1 \quad , \quad n > 0 \quad , \quad t_{m,n} \leq t_{m+1,0} \tag{1}$$

$t_{m,0}$ is the time of the m'th postsynaptic spike and $t_{m,n}$ $(n > 0)$ is the time of the n'th presynaptic spike after the m'th postsynaptic spike. The weight changes when the cell fires an action potential:

$$w(t_{m,0}) = w(t_{m-1,0}) + \alpha e^{-\frac{t_{m,0} - t_{m-1,s}}{\tau}} c(t_{m-1,s}) - \beta w(t_{m-1,0}) \qquad (2)$$
$$s = max\{v : t_{m-1,v} \leq t_{m,0}\}$$

where $w$ is the weight at this synapse. $t_{m-1,s}$ means the last event (presynaptic or postsynaptic spike) before the m'th postsynaptic spike. $\alpha$ and $\beta$ are parameters influencing learning speed and weight vector normalization (see (5)).

Our learning rule is designed to react to temporal correlations between spikes in the input signals. However, to show the normalizing of the weights we analyze its behavior by making some simplifying assumptions on the input and output signals; e.g. the intervals of the presynaptic and the postsynaptic spike train are Poisson distributed and there is no correlation between single spikes. Therefore we can represent the signals by their instantaneous average frequencies $O$ and $\vec{I}$. Now the simplified learning rule can be written as:

$$\frac{\delta}{\delta t}\vec{w} = \alpha l(O)\vec{I} - \beta \vec{w} O \qquad (3)$$

$$l(O) = O\tau(1 - e^{-\frac{1}{O\tau}}) \qquad (4)$$

$l(O)$ represents the average percentage to which the correlation signal is reduced between weight updates (output spikes). So when the neuron's average firing rate fulfills $O \gg \frac{1}{\tau}$, one can approximate $l(O) \approx 1$. (3) is thus reduced to the Riccati equation described by Kohonen [1]. This rule would not be Hebbian, but normalizes the weight vector (see (5)). Note that if the correlation signal does not decay, then our rule matches exactly the Riccati equation. We will further refer to it as the Modified Riccati Rule (MRR). Whereas if $O \ll \frac{1}{\tau}$ then $l(O) \approx O\tau$, which is a Hebbian learning rule also described in [1].

If we assume that the spiking mechanism preserves $O = \vec{w}^T \vec{I}$ and insert it in (3), it follows for the equilibrium state:

$$\|\vec{w}\| = \sqrt{l(O)\frac{\alpha}{\beta}} \qquad (5)$$

Since $l(O) < 1$ the weight vector will never be longer than $\sqrt{\frac{\alpha}{\beta}}$. This property also holds when the simplifying assumptions are removed. The vector will always be smaller, as it is with no decay of the correlation signals, since the decay only affects the incrementing part of the rule.

Matters get much more complicated with the removal of the assumption of the pre- and postsynaptic trains being independently Poisson distributed. With an integrate-and-fire neuron for instance, or if there exist correlations between spikes of the input trains, it is no longer possible to express what happens in terms of rate

coding only (with $\vec{I}$ and $O$). (3) is still valid as an approximation but temporal relationships between pre- and postsynaptic spikes become important. Presynaptic spikes immediately followed by an action potential will have the strongest increasing effect on the synapse's weight.

## 3  IMPLEMENTATION IN ANALOG VLSI

We have implemented a learning rule in a neuron circuit fabricated in a $2.0\mu m$ CMOS process. This neuron is a preliminary design that conforms only approximately to the MRR. The neuron uses an integrate-and-fire mechanism to generate action potentials (Figure 2).

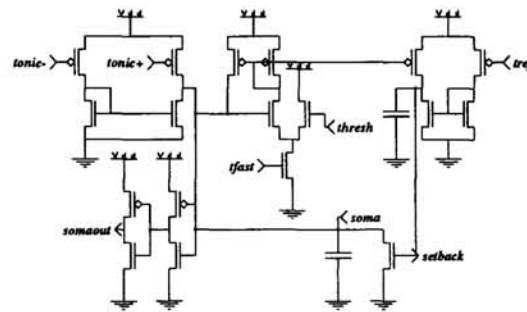

Figure 2: Integrate-and-fire neuron. The *soma* capacitor holds the somatic membrane voltage. This voltage is compared to a threshold *thresh* with a differential pair. When it crosses this threshold it gets pulled up through the mirrored current from the differential pair. This same current gets also mirrored to the right and starts to pull up a second leaky capacitor (*setback*) through a small $W/L$ transistor, so this voltage rises slowly. This capacitor voltage finally opens a transistor that pulls *soma* back to ground where it restarts integrating the incoming current. The parameters *tonic+* and *tonic−* are used to add or subtract a constant current to the soma capacitor. *tref* allows the spike-width to be changed.

Not shown, but also part of the neuron, are two non-learning synapses: one excitatory and one inhibitory. Each of three learning synapses contains a storage capacitor for the synaptic weight and for the correlation signal (Figure 3).

The correlation signal $c$ is simplified to a binary variable in this implementation. When an input spike occurs, the correlation signal is set to 1. It is set to 0 whenever the neuron produces an output-spike or after a fixed time-period ($T$ in (7)) if there is no other input spike:

$$c(t_{m,0}) = 0$$
$$c(t_{m,n}) = 1 \quad , \quad n > 0 \quad , \quad t_{m,n} \le t_{m+1,0} \tag{6}$$

This approximation unfortunately tends to eliminate differences between highly active inputs and weaker inputs. Nevertheless the weight changes with every output spike:

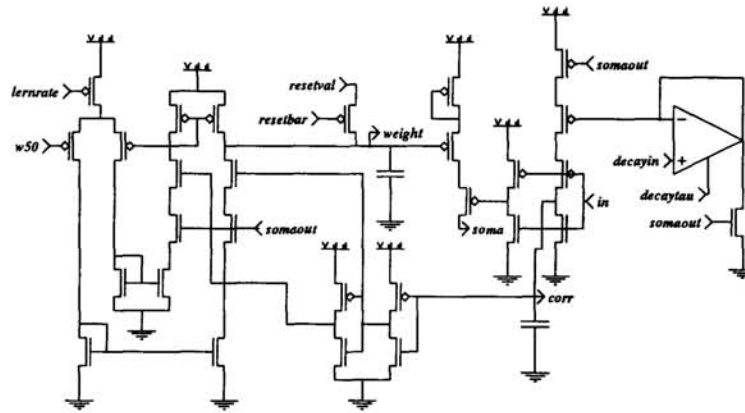

Figure 3: The CMOS learning-synapse incorporates the learning mechanism. The *weight* capacitor holds the weight, the *corr* capacitor stores the correlation signal representation. The magnitude of the weight increment and decrement are computed by a differential pair (upper left $w50$). These currents are mirrored to the synaptic weight and gated by digital switches encoding the state of the correlation signal and of the somatic action potential. The correlation signal reset is mediated by a leakage transistor, *decayin*, which has a tonic value, but is increased dramatically when the output neuron fires.

$$w(t_{m,0}) = w(t_{m-1,0}) \begin{cases} +\alpha \frac{e^{w50}}{e^{w50}+e^{w(t_{m-1,0})}} & \text{if } c(t_{m-1,s}) = 1 \text{ and } t_{m,0} - t_{m-1,s} < T \\ -\alpha \frac{e^{w(t_{m-1,0})}}{e^{w50}+e^{w(t_{m-1,0})}} & \text{otherwise} \end{cases}$$

$$s = max\{v : t_{m-1,v} \le t_{m,0}\} \tag{7}$$

$w$ is the weight on one synapse, $c$ is the correlation signal of that synapse, and $\alpha$ is a parameter that controls how fast the weight changes. (See in the previous section for a description of $t_{m,n}$.) The weight, $w_{50}$, is the equilibrium value of the synaptic weight when the occurrence of an input spike is fifty percent correlated with the occurrence of an output spike. This implementation differs from the Riccati rule in that either the weight increment or the weight decrement, but not both, are executed upon each output spike. Also, the weight increment is a function of the synaptic weight. The circuit was implemented this way to try and achieve an equilibrium value for the synaptic weight equal to the fraction of the time that the input neuron fired relative to the times the output neuron fired. This is the correct equilibrium value for the synaptic weight in the Riccati rule. The evolution of a synaptic weight is depicted in Figure 4.

The synaptic weight vector normalization in this implementation is accurate only when the assumptions of the design are met. These assumptions are that there is one or fewer input spikes per synapse for every output spike. This assumption is easier to meet when there are many synapses formed with the neuron, so that spikes from multiple inputs combine to drive the cell to threshold. Since we have only three synapses, this approximation is usually violated. Nevertheless, the weights compete with one another and therefore the length of the weight vector is limited. Competition between synaptic weights occurs because if one weight is stronger, it causes the output neuron to spike and this suppresses the other input that has not

fired. Future revision of the chip will conform more closely to the MRR.

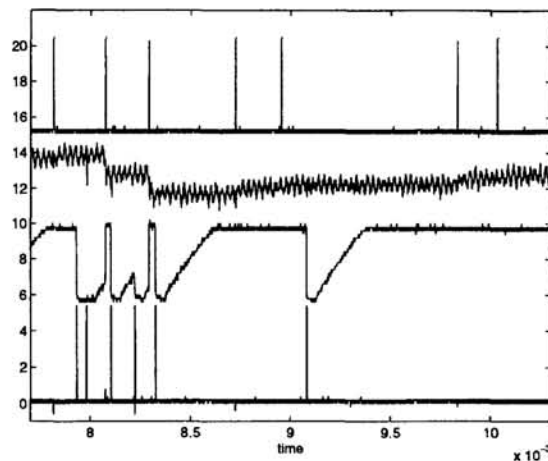

Figure 4: A snapshot of the learning behavior of a single VLSI synapse: The top trace is the neuron output (1V/division), the upper middle trace is the synaptic weight (lower voltage means a stronger synaptic weight) (25mV/division), the lower middle trace is a representation of the correlation signal (1 V/division)(it has inverted sense too) and the bottom trace is the presynaptic activity (1V/division). The weight changes only when an output spike occurs. The timeout of the correlation signal is realized with a decay and a threshold. If the correlation signal is above threshold, the weight is strengthened. If the signal has decayed below threshold at the time of an output spike, the weight is weakened. The magnitude of the change of the weight is a function of the absolute magnitude of the weight. This weight was weaker than $w_{50}$, so the increments are bigger than the decrements.

## 4  TEMPORAL CORRELATION IN INPUT SPIKE TRAINS

Figure 5 illustrates the ability of our learning rule to detect temporal correlations in spike trains. A simulated neuron strengthens those two synapses that receive 40% coincident spikes, although all four synapses get the same average spike frequencies.

## 5  DISCUSSION

Learning rules that make use of temporal correlations in their spike inputs/outputs provide biologically relevant mechanisms of synapse modification [5, 7, 6]. Analog VLSI implementations allow such models to operate in real time. We plan to develop such analog VLSI neurons using floating gates for weight storage and an address-event bus for interneuronal connections. These could then be used in realtime applications in adaptive 'neuromorphic' systems.

**Acknowledgments**

We thank the following organizations for their support: SPP Neuroinformatik des Schweizerischen Nationalfonds, Centre Swiss d'Electronique et de Microtechnique, U.S. Office of Naval Research and the Gatsby Charitable Foundation.

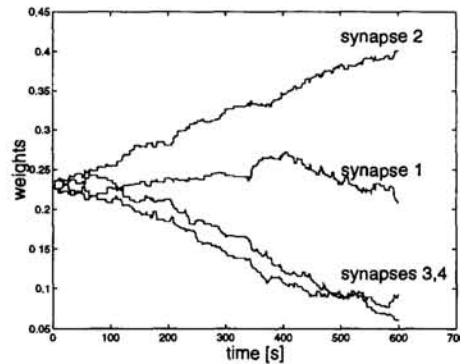

Figure 5: In this simulation we use a neuron with four synapses. All of them get input trains of the same average frequency (20Hz). Two of those input trains are the result of independent Poisson processes (synapses 3 and 4), the other two are the combination of two Poisson processes (synapses 1 and 2): One that is independent of any other (12Hz) and one that is shared by the two with slightly different time delays (8Hz): Synapse 1 gets those coincident spikes 0.01 seconds earlier than synapse 2. Synapse 2 gets stronger because when it together with synapse 1 triggered an action potential, it was the last synapse being active before the postsynaptic spike. The parameters were: $\alpha = 0.004, \beta = 0.02, \tau = 11ms$

# References

[1] Tuevo Kohonen. *Self-Organization and Associative Memory.* Springer, Berlin, 1984.

[2] D.K. Ferry L.A. Akers and R.O. Grondin. Synthetic neural systems in the 1990s. *An introduction to neural and electronic networks, Academic Press (Zernetzer, Davis, Lau, McKenna)*, pages 359–387, 1995.

[3] J. Lazzaro, J. Wawrzynek, M. Mahowald, M. Sivilotti, and D. Gillespie. Silicon auditory processors as computer peripherals. *IEEE Trans. Neural Networks*, 4:523–528, 1993.

[4] A. Mortara and E. A. Vittoz. A communication architecture tailored for analog VLSI artificial neural networks: intrinsic performance and limitations. *IEEE Translation on Neural Networks*, 5:459–466, 1994.

[5] G. J. Stuart and B. Sakmann. Active propagation of somatic action potentials into neocortical pyramidal cell dendrites. *Nature*, 367:69ff, 1994.

[6] M. V. Tsodyks and H. Markram. Redistribution of synaptic efficacy between neocortical pyramidal neurons. *Nature*, 382:807–810, 1996.

[7] R. Yuste and W. Denk. Dendritic spines as basic functional units of neuronal integration. *Nature*, 375:682–684, 1995.
